# Temporal Low-Order Statistics of Natural Sounds

**H. Attias\* and C.E. Schreiner†**
Sloan Center for Theoretical Neurobiology and
W.M. Keck Foundation Center for Integrative Neuroscience
University of California at San Francisco
San Francisco, CA 94143-0444

## Abstract

In order to process incoming sounds efficiently, it is advantageous for the auditory system to be adapted to the statistical structure of natural auditory scenes. As a first step in investigating the relation between the system and its inputs, we study low-order statistical properties in several sound ensembles using a filter bank analysis. Focusing on the amplitude and phase in different frequency bands, we find simple parametric descriptions for their distribution and power spectrum that are valid for very different types of sounds. In particular, the amplitude distribution has an exponential tail and its power spectrum exhibits a modified power-law behavior, which is manifested by self-similarity and long-range temporal correlations. Furthermore, the statistics for different bands within a given ensemble are virtually identical, suggesting translation invariance along the cochlear axis. These results show that natural sounds are highly redundant, and have possible implications to the neural code used by the auditory system.

## 1 Introduction

The capacity of the auditory system to represent the auditory scene is restricted by the finite number of cells and by intrinsic noise. This fact limits the ability of the organism to discriminate between different sounds with similar spectro-temporal

characteristics. However, it is possible to enhance the discrimination ability by a suitable choice of the encoding procedure used by the system, namely of the transformation of sounds reaching the cochlea to neural spike trains generated in successive processing stages in response to these sounds. In general, the choice of a good encoding procedure requires knowledge of the statistical structure of the sound ensemble.

For the visual system, several investigations of the statistical properties of image ensembles and their relations to neuronal response properties have recently been performed (Field 1987, Atick and Redlich 1990, Ruderman and Bialek 1994). In particular, receptive fields of retinal ganglion and LGN cells were found to be consistent with an optimal-code prediction formulated within information theory (Atick 1992, Dong and Atick 1995), suggesting that the visual periphery may be designed as to take advantage of simple statistical properties of visual scenes.

In order to investigate whether the auditory system is similarly adapted to the statistical structure of its own inputs, a good characterization of auditory scenes is necessary. In this paper we take a first step in this direction by studying low-order statistical properties of several sound ensembles. The quantities we focus on are the spectro-temporal amplitude and phase defined as follows. For the sound $s(t)$, let $s_\nu(t)$ denote its components at the set of frequencies $\nu$, obtained by filtering it through a bandpass filter bank centered at those frequencies. Then

$$s_\nu(t) = x_\nu(t) \cos\left(\nu t + \phi_\nu(t)\right) \tag{1}$$

where $x_\nu(t) \geq 0$ and $\phi_\nu(t)$ are the spectro-temporal amplitude (STA) and phase (STP), respectively. A complete characterization of a sound ensemble with respect to a given filter bank must be given by the joint distribution of amplitudes and phases at all times, $p\left(x_{\nu_1}(t_1), \phi_{\nu_1}(t'_1), ..., x_{\nu_n}(t_n), \phi_{\nu_n}(t'_n)\right)$. In this paper, however, we restrict ourselves to second-order statistics in the time domain and examine the distribution and power spectrum of the stochastic processes $x_\nu(t)$ and $\phi_\nu(t)$.

Note that the STA and STP are quantities directly relevant to auditory processing. The different stages of the auditory system are organized in topographic frequency maps, so that cells tuned to the same sound frequency $\nu$ are organized in stripes perpendicular to the direction of frequency progression (see, e.g., Pickles 1988). The neuronal responses are thus determined by $x_\nu$ and $\phi_\nu$, and by $x_\nu$ alone when phase-locking disappears above 4–5KHz.

## 2    Methods

Since it is difficult to obtain a reliable sample of an animal's auditory scene over a sufficiently long time, we chose instead to analyze several different sound ensembles, each consisting of a 15min sound of a certain type. We used cat vocalizations, bird songs, wolf cries, environmental sounds, symphonic music, jazz, pop music, and speech. The sounds were obtained from commercially available compact discs and from recordings of animal vocalizations in two laboratories. No attempt has been made to manipulate the recorded sounds in any way (e.g., by removing noise).

Each sound ensemble was loaded into the computer by 30sec segments at a sampling rate of $f_s = 44.1$KHz. After decimating to $f_s/2$, we performed the following frequency-band analysis. Each segment was passed through a bandpass fil-

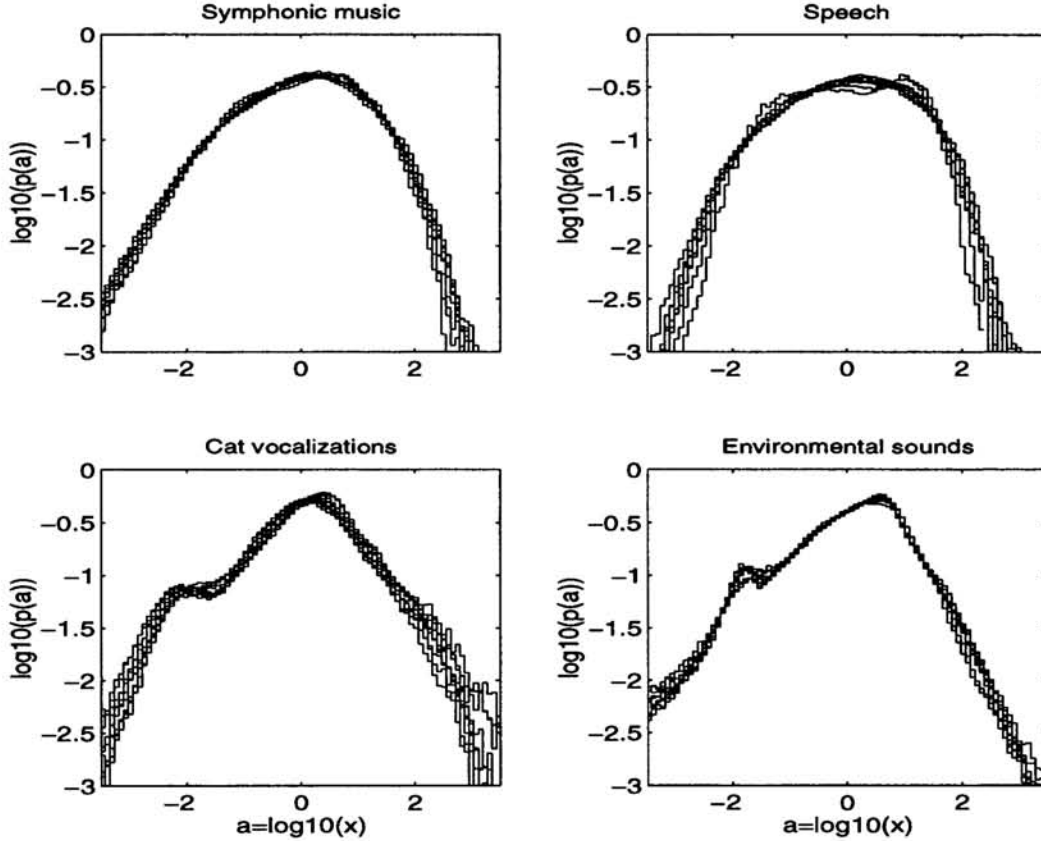

Figure 1: Amplitude probability distribution in different frequency bands for four sound ensembles.

ter bank with impulse responses $h_\nu(t)$ to get the narrow-band component signals $s_\nu(t) = s(t) * h_\nu(t)$. We used square, non-overlapping filters with center frequencies $\nu$ logarithmically spaced within the range of $100 - 11025\text{Hz}$. The filters were usually 1/8-octave wide, but we experimented with larger bandwidths as well. The amplitude and phase in band $\nu$ were then obtained via the Hilbert transform

$$H\left[s_\nu(t)\right] = s_\nu(t) + \frac{i}{\pi} \int dt' \frac{s(t')}{t - t'} = x_\nu(t)e^{i(\nu t + \phi_\nu(t))} \ . \qquad (2)$$

The frequency content of $x_\nu$ is bounded by 0 and by the bandwidth of $h_\nu$ (Flanagan 1980), so keeping the latter below $\nu$ guarantees that the low frequencies in $s_\nu$ are all contained in $x_\nu$, confirming its interpretation as the amplitude modulator of the carrier $\cos \nu t$ suggested by (1). The phase $\phi_\nu$, being time-dependent, produces frequency modulation. For a given $\nu$ the results were averaged over all segments.

## 3   Amplitude Distribution

We first examined the STA distribution in different frequency bands $\nu$. Fig. 1 presents historgrams of $p(\log_{10} x_\nu)$ on a logarithmic scale for four different sound ensembles. In order to facilitate a comparison among different bands and ensembles, we normalized the variable to have zero mean and unit variance, $\langle \log_{10} x_\nu(t) \rangle = 0$, $\langle (\log_{10} x_\nu(t))^2 \rangle = 1$, corresponding to a linear gain control.

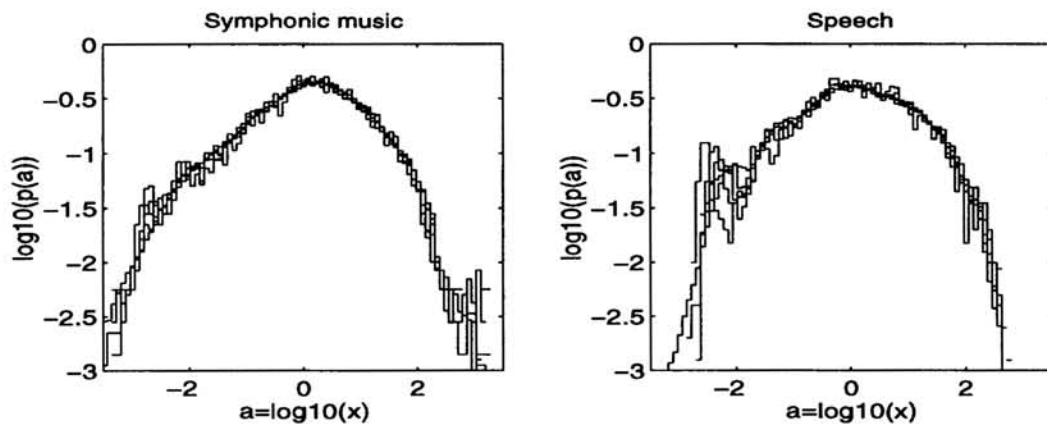

Figure 2: $n$-point averaged amplitude distributions for $\nu = 800$Hz in two sound ensembles, using $n = 1, 20, 50, 100, 200$. The speech ensemble is different from the one used in Fig. 1.

As shown in the figure, within a given ensemble, the histograms corresponding to different bands lie atop one another. Furthermore, although curves from different ensembles are not identical, we found that they could all be fitted accurately to the same parametric functional form, given by

$$p(x_\nu) \propto \frac{e^{-\gamma x_\nu}}{(b_0^2 + x_\nu^2)^{\beta/2}} \qquad (3)$$

with parameter values roughly in the range of $0.1 \leq \gamma \leq 1$, $0 \leq \beta \leq 2.5$, and $0.1 \leq b_0 \leq 0.6$. In some cases, a mixture of two distributions of the form (3) was necessary, suggesting the presence of two types of sound sources; see, e.g., the slight bimodality in the lower parts of Fig. 1. Details of the fitting procedure will be given in a longer paper. We found the form (3) to be preserved as the filter bandwidths increased.

Whereas this distribution decays exponentially fast at high amplitudes ($p \propto e^{-\gamma x_\nu}/x_\nu^\beta$), it does not vanish at low amplitudes, indicating a finite probability for the occurence of arbitrarily soft sounds. In contrast, the STA of a Gaussian noise signal can be shown to be distributed according to $p \propto x_\nu e^{-\lambda x_\nu^2}$, which vanishes at $x_\nu = 0$ and decays faster than (3) at large $x_\nu$. Hence, the origin of the large dynamic range usually associated with audio signals can be traced to the abundance of soft sounds rather than of loud ones.

## 4  Amplitude Self-Similarity

An interesting probe of the STA temporal correlations is the property of scale invariance (also called statistical self-similarity). The process $x_\nu(t)$ is scale-invariant when any statistical quantity on a given scale (e.g., at a given temporal resolution, determined by the sampling rate) does not change as that scale is varied. To observe this property we examined the STA distribution $p(x_\nu)$ at different temporal resolutions, by defining the $n$-point averaged amplitude

$$x_\nu^{(n)}(t) = \frac{1}{n}\sum_{k=0}^{n-1} x_\nu(t + k\Delta) \qquad (4)$$

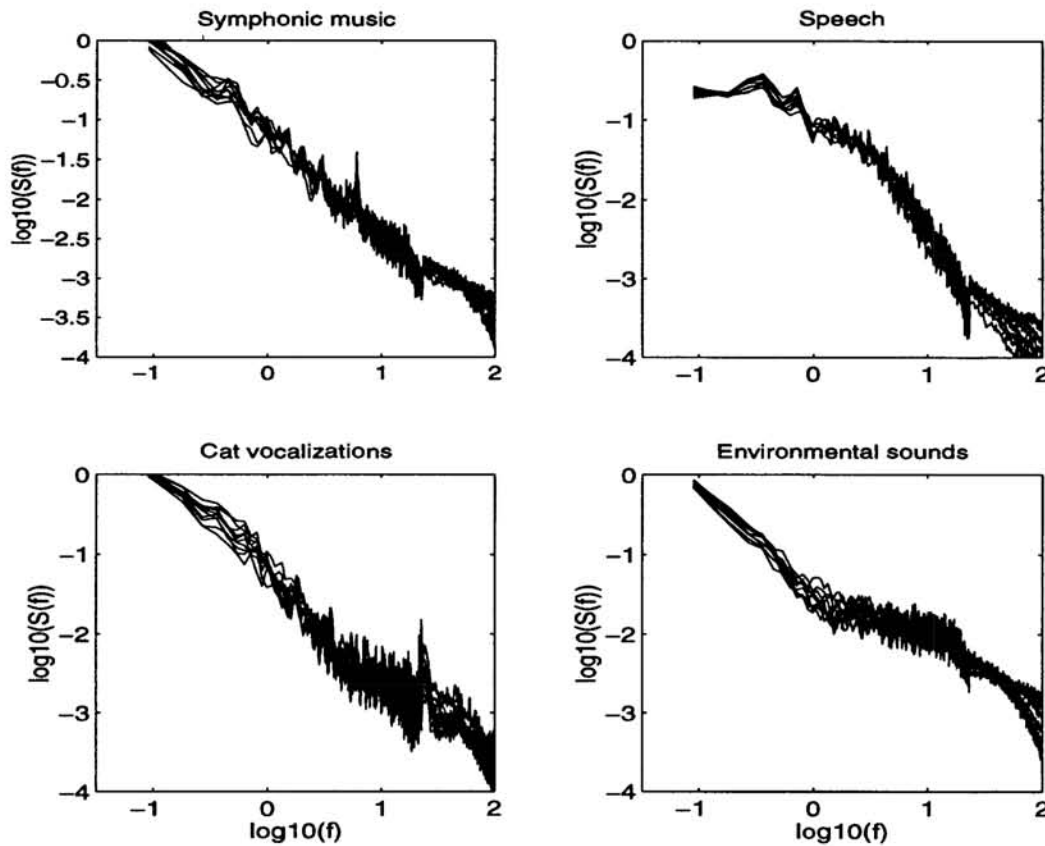

Figure 3: Amplitude power spectrum in different frequency bands for four sound ensembles.

($\Delta = 1/f_s$) and computing its distribution. Fig. 2 displays the histograms of $p(\log_{10} x_\nu^{(n)})$ for the $\nu = 800$Hz frequency band in two sound ensembles on a logarithmic scale, using $n = 1, 20, 50, 100, 200$ which correspond to a temporal resolution range of $0.75 - 150$msec. Remarkably, the histogram remains unmodified even for $n = 200$. Had the $x_\nu(t + k\Delta)$ been statistically independent variables, the central limit theorem would have predicted a Gaussian $p(x_\nu^{(n)})$ for large $n$. The fact that this non-Gaussian distribution preserves its form as $n$ increases implies the presence of temporal STA correlations over long periods.

Notice the analogy between the invariance of $p(x_\nu)$ under a change in filter bandwidth, reported in the previous section, and under a change in temporal resolution. An $x_\nu$ with a broad bandwidth is essentially an average over the $x_\nu$'s with narrow bandwidth within the same band, thus bandwidth invariance is a manifestation of STA correlations across frequency bands.

## 5 Amplitude Power Spectrum

In order to study the temporal amplitude correlations directly, we computed the STA power spectrum $S_\nu(\omega) = \langle |\, \tilde{x}_\nu(\omega) \,|^2 \rangle$ in different bands $\nu$, where $\tilde{x}_\nu(\omega)$ is the Fourier transform of the log-amplitude $\log_{10} x_\nu(t)$ obtained by a 512-point FFT. As is well-known, the spectrum $S_\nu(\omega)$ is the Fourier transform of the log-amplitude auto-correlation function $c_\nu(\tau) = \langle \log_{10} x_\nu(t) \log_{10} x_\nu(t + \tau) \rangle$. We used

the zero-mean, unit-variance normalization of $\log_{10} x_\nu$, which implies the normalization $\int d\omega S_\nu(\omega) = $ const. of the spectra. Fig. 3 presents $S_\nu$ as a function of the modulation frequency $f = \omega/2\pi$ on a logarithmic scale for four different sound ensembles. Notice that, as in the case of the STA distribution, the different curves corresponding to different frequency bands within a given ensemble lie atop one another, including individual peaks; and whereas spectra in different ensembles are not identical, we found a simple parametric description valid for all ensembles which is given by

$$S_\nu(\omega) \propto \frac{1}{(\omega_0^2 + \omega^2)^{\alpha/2}} , \tag{5}$$

with parameter values roughly in the range of $1 \leq \alpha \leq 2.5$ and $10^{-4} \leq \omega_0 \leq 1$. This is a modified power-law form (note that $S_\nu \to C/\omega^\alpha$ at large $\omega$), implying long-rangle temporal correlations in the amplitude: these correlations decrease slowly (as a power law in $t$) on a time scale of $1/\omega_0$, beyond which they decay exponentially fast. Larger $\omega_0$ contributes more to the flattening of the spectrum at low frequencies (see especially the speech spectra) and corresponds to a shorter correlation time. Again, in some cases a sum of two such forms was necessary, corresponding to a mixture STA distribution as mentioned above; see, e.g., the environmental sound spectra (lower right part of Fig. 3 and Fig. 1).

The form (5) persisted as the filter bandwidth increased. In the limit of allpass filter (not shown) we still observed this form, a fact related to the report of (Voss and Clarke 1975) on $1/f$-like power spectra of sound 'loudness' $s(t)^2$ found in several speech and music ensembles.

## 6 Phase Distribution and Power Spectrum

Whereas the STA is a non-stationary process which is locally stationary and can thus be studied on the appropriate time scale using our methods, the STP is non-stationary even locally. A more suitable quantity to examine is its rate of change $d\phi_\nu/dt$, called the instantaneous frequency. We studied the statistics of $|\, d\phi_\nu/dt \,|$ in different ensembles, and found its distribution to be described accurately by the parametric form (3) with $\gamma = 0$, whereas its power spectrum could be well fitted by the form (5). In addition, those quantities were virtually identical in different bands within a given ensemble. More details on this work will be provided in a longer paper.

## 7 Implications for Auditory Processing

We have shown that auditory scenes have several robust low-order statistical properties. The STA power spectrum has a modified power-law behavior, which is manifested in self-similarity and temporal correlations over a few hundred milliseconds. The distribution has an exponential tail and features a finite probability for arbitrarily soft sounds. Both the phase and amplitude statistics can be described by simple parametrized functional forms which are valid for very different types of sounds. These results lead to the conclusion that natural sounds are highly redundant, i.e., they occupy a very small subspace in the space of all possible sounds. It would therefore be beneficial for the auditory system to adapt its sound representation to these statistics, thus improving the animal discrimination ability. Whether

the auditory system actually follows this design principle is an empirical question which can be attacked by suitable experiments.

Furthermore, since different frequency bands correspond to different spatial locations on the basal membrane (Pickles 1988), the fact that the distributions and spectra in different bands within a given ansemble are identical suggests the existence of translation invariance along the cochlear axis, i.e., all the locations in the cochlea 'see' the same statistics. This is analogous to the translation invariance found in natural images.

Finally, a recent theory for peripheral visual processing (Dong and Atick 1995) proposes that, in order to maximize information transmission into cortex, the LGN performs temporal correlation of retinal images. Within an analogous auditory model, the decorrelation time for sound ensembles reported here implies that the auditory system should process incoming sounds by a few hundred msec-long segments. The ability of cortical neurons to follow in their response modulation rates near and below 10Hz but usually not higher (Schreiner and Urbas 1988) may reflect such a process.

## Acknowledgements

We thank B. Bonham, K. Miller, S. Nagarajan, and especially W. Bialek for helpful discussions and suggestions. We also thank F. Theunissen for making his bird song recordings available to us. Supported by The Office of Naval Research (N00014-94-1-0547). H.A. was supported by a Sloan Foundation grant for the Sloan Center for Theoretical Neurobiology.

## Footnotes

\*Corresponding author. E-mail: hagai@phy.ucsf.edu.

†E-mail: chris@phy.ucsf.edu.

## References

J.J. Atick and N. Redlich (1990), Towards a theory of early visual processing. Neural Comput. **2**, 308-320.

J.J. Atick (1992), Could information theory provide an ecological theory of sensory processing. Network **3**, 213-251.

D.W. Dong and J.J. Atick (1995), Temporal decorrelation: a theory of lagged and non-lagged responses in the lateral geniculate nucleus. Network **6**, 159-178.

D.J. Field (1987), Relations between the statistics of natural images and the response properties of cortical cells. J. Opt. Soc. Am. **4**, 2379-2394.

J.L. Flanagan (1980), Parametric coding of speech spectra. J. Acoust. Soc. Am. **68**, 412-419.

J.O. Pickles (1988), *An introduction to the physiology of hearing* (2nd Ed.). San Diego, CA: Academic Press.

D.L. Ruderman and W. Bialek (1994), Statistics of natural images: scaling in the woods. Phys. Rev. Lett. **73**, 814-817.

C.E. Schreiner and J.V. Urbas, Representation of amplitude modulation in the auditory cortex of the cat. II. Comparison between cortical fields. Hear. Res. **32**, 49-63.

R.F. Voss and J. Clarke (1975), $1/f$ noise in music and speech. Nature **258**, 317-318.